# Semi-Open 3D Object Retrieval via Hierarchical Equilibrium on Hypergraph

**Yang Xu**[1], **Yifan Feng**[1], **Jun Zhang**[2], **Jun-Hai Yong**[1], **and Yue Gao**[1]*

[1]BNRist, THUIBCS, KLISS, BLBCI, School of Software, Tsinghua University, China
[2]Tencent AI Lab
{xuyang9610,evanfeng97}@gmail.com, junejzhang@tencent.com,
{yongjh,gaoyue}@tsinghua.edu.cn

## Abstract

Existing open-set learning methods consider only the single-layer labels of objects and strictly assume no overlap between the training and testing sets, leading to contradictory optimization for superposed categories. In this paper, we introduce a more practical ***Semi-Open Environment*** setting for open-set 3D object retrieval with hierarchical labels, in which the training and testing set share a partial label space for coarse categories but are completely disjoint from fine categories. We propose the Hypergraph-Based Hierarchical Equilibrium Representation (HERT) framework for this task. Specifically, we propose the Hierarchical Retrace Embedding (HRE) module to overcome the global disequilibrium of unseen categories by fully leveraging the multi-level category information. Besides, tackling the feature overlap and class confusion problem, we perform the Structured Equilibrium Tuning (SET) module to utilize more equilibrial correlations among objects and generalize to unseen categories, by constructing a superposed hypergraph based on the local coherent and global entangled correlations. Furthermore, we generate four semi-open 3DOR datasets with multi-level labels for benchmarking. Results demonstrate that the proposed method can effectively generate the hierarchical embeddings of 3D objects and generalize them towards semi-open environments.

## 1 Introduction

3D objects are of paramount significance, finding extensive applications from computer graphics [1] to security [31] and autonomous robotics [2]. As the fundamental task of data acquisition, 3D object retrieval (3DOR) [22, 10] plays a pivotal role in the computer vision community [3]. 3DOR methods learn to represent 3D objects from the training set and then extract features from query objects to effectively align similar samples. According to the category overlap between training and testing sets, existing 3DOR algorithms can be divided into closed-set and open-set types. The former conducts retrieval for objects whose categories have been seen in the training set [32, 36], while the latter handles objects of unseen categories [7].

Existing open-set 3D learning methods are based on the assumption that the labels of object categories are at a single level [46, 23]. In practical scenarios, objects are typically described by multiple hierarchical labels. This leads to categories in training and testing sets showing varying degrees of overlap at different levels. As illustrated in Figure 1(a), we term this scenario, where the training and testing set share a partial label space of coarse categories but are completely disjoint from fine categories, as ***Semi-Open Environment***. Besides, existing methods typically extract basic features using pre-trained models, followed by further open-set learning and optimization. However, for

---

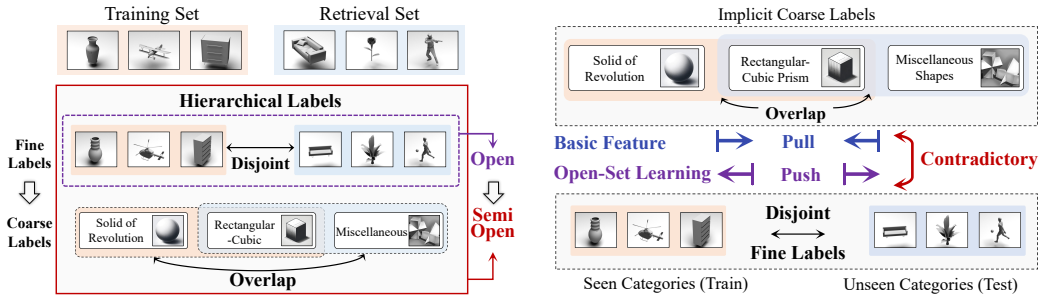

(a) Hierarchical labels in semi-open environment.　　(b) Optimization contradiction of hierarchical labels.

Figure 1: Illustration of motivation from open-set to semi-open 3DOR. Objects may be more accurately described by hierarchical labels than single-level labels in real-world scenarios. In this semi-open setting, the training and testing set share a partial label space for coarse categories but are completely disjoint from fine categories.

categories with the same coarse label but different fine labels, the training of the basic model and the open-set learning will result in contradictory optimizations as shown in Figure 1(b).

Specifically for 3D object retrieval techniques, only a few methods [7, 32] have been explored for open-set retrieval. These methods consider only the single-layer labels of objects and strictly assume no overlap between the training and testing data distributions. The existing open-set 3DOR method [7] treats objects from seen and unseen categories as isomorphic vertices and constructs a graph model based on global correlations. On one hand, this isomorphic model overlooks hierarchical correlations inherent in multi-level categories, particularly the shared coarse labels. Consequently, the embedding distribution from structure-aware learning tends to be unbalanced towards the feature space of seen categories in the training set. On the other hand, this structure focuses on global correlations between seen and unseen categories, neglecting intricate local correlations within unseen categories themselves. This lack of local attention may result in issues like feature overlap and class confusion within these unseen categories.

Focusing on this practical semi-open environment for retrieval, where the training and testing set share a partial label space for coarse categories but are completely disjoint from fine categories, we introduce the semi-open 3D object retrieval task and construct four datasets with multi-level labels to expand the application of 3DOR. We propose the Hypergraph-Based Hierarchical Equilibrium Representation (HERT) framework for semi-open 3DOR. To overcome the global disequilibrium of unseen categories, we propose the Hierarchical Retrace Embedding module (HRE) to achieve balanced representation across multi-level categories. This module generates multi-level retrace embeddings for capturing the hierarchical semantics of objects. To tackle the feature overlap and class confusion problem, we propose the Structured Equilibrium Tuning (SET) module. This module utilizes high-order correlations among objects for unseen category generalization, by constructing a superposed Hypergraph based on local coherent and global entangled correlations. In summary, our main contributions are fourfold:

- We introduce the semi-open 3D object retrieval task to refine the setting of the 3D object retrieval task in real-world scenario applications, and we construct four datasets for benchmarking downstream tasks.
- We propose the Hypergraph-Based Hierarchical Equilibrium Representation (HERT) framework for the semi-open 3DOR task, including the Hierarchical Retrace Embedding (HRE) and the Structured Equilibrium Tuning (SET) modules, which are designed to overcome the distribution disequilibrium and confusion of unseen categories.
- We propose a superposed Hypergraph structure to capture high-order correlations among objects, under the guidance of local coherent correlations and global entangled correlations from hierarchical category information.
- Experimental results on the four datasets demonstrate that our method can outperform state-of-the-art retrieval methods towards the semi-open environment.

## 2 Related Work

**3D Object Retrieval.** 3D object retrieval (3DOR) aims to find the relevant objects from the target set for the query objects, most 3DOR methods construct feature alignment models for 3D objects through metric learning. MMJN [20] proposes a discrimination loss to minimize the distance between objects belonging to the same categories, and learn discriminative embeddings for retrieval. MIFN [17] constructs fusion networks by weighted concatenation for modality-specific features. PVNet [42] proposes a joint network for the fusion of multi-view and point cloud features. PVRNet [43] proposes an attention mechanism to generate the unified embedding of different modalities. CMCL [14] designs a cross-modal center loss to compress features of different modalities to a modal-invariant space. However, most existing methods mainly focus on close-set retrieval. Currently, only a few methods [7, 32, 39, 38, 37] have explored open-set retrieval, but they typically assume no overlap between training and testing data distributions, which is at odds with the semi-open setting.

**Open-Set Learning.** Open-set learning (OSL) methods can be roughly separated into two categories [45], *i.e.*, discriminative methods and generative models. In discriminative models, traditional methods achieve the classification of unknown and known categories based on the Support Vector Machine (SVM) [27, 28] or the Extreme Value Theory (EVT) [12, 26]. Recently, deep learning methods for OSL have made remarkable rapid progress. OpenMax [4] is the first algorithm proposed to replace the SoftMax layer and calibrate the output probability with the Weibull distribution. Then, [29] and [41] utilize the one-vs-rest units and reconstructed latent representation for unknown detection. Generative OSL models are designed to anticipate the distribution of novel classes through training. PROSER [45] allocates placeholders for both data and classifier to detect the unknown classes, C2AE [21] propose a two-step framework to tackle open-set recognition problem with close-set training and open-set training, respectively. Both types of open-set learning methods [9, 19] are based on the assumption that there is no overlap between known and unknown data distribution, thus facilitating the design of classifiers for unseen object detection, but are challenging to generalize to practical scenarios that involve hierarchical superposed categories.

**Multi-Label Learning.** Multi-label learning aims to create a model that can assign multiple labels for each instance simultaneously. Existing multi-label learning methods are mostly designed for recognition tasks, and based on one-vs-all classifiers [40, 24] and embeddings [44, 18]. As for the retrieval task, DMSSPH [33] proposes a multi-level preserving hashing network, and AMD-GCN [16] designs a GCN-based network for multi-label pattern image retrieval. TranGCN [15] proposes a cross-modal attention mechanism at each layer for multi-label embeddings. Although these methods construct effective models for image embedding, they overlook the hierarchical relationship and dependencies between different categories. In this paper, we aim to model the correlations between hierarchical label information and generalize to unseen categories.

## 3 Problem Setup

### 3.1 3D Object Retrieval

The goal of the 3D object retrieval (3DOR) is to develop a method using the training set $\mathcal{S}_{trn} = \{(o_i, y_i^c, y_i^f)\}_{i=1}^L$, which is then employed to identify similar objects in the retrieval (testing) set $\mathcal{S}_{ret} = \{(o_i, \hat{y}_i^c, \hat{y}_i^f)\}_{i=1}^R$, which is comprised of the query set $\mathcal{S}_q$ and the target set $\mathcal{S}_t$. Here, $L$ and $R$ represent the number of samples in the training and testing (retrieval) sets, respectively. The expressions $y_i^f \in \mathcal{Y}^f = \{c_j^f\}_{j=1}^{Y^f}$ and $y_i^c \in \mathcal{Y}^c = \{c_k^c\}_{k=1}^{Y^c}$ denote the fine and coarse category labels of 3D object $o_i$, respectively. Typically, $y_i^c$ provides the category label from a more general (coarse) level, such as the basic geometric shapes or other common attributes between $\mathcal{S}_{trn}$ and $\mathcal{S}_{ret}$, whereas $y_i$ provides a semantic-specific (fine) category label. Generally, the number of coarse categories $Y^c$ is much smaller than the number of fine categories, which is $Y^c \ll Y^f$. For 3D objects, The $o_i = \{m_r\}_{r=1}^M$ denotes the representation by $M$ modalities, *i.e.*, multi-view, point cloud and voxel.

### 3.2 Semi-Open 3D Object Retrieval

In traditional open-set 3DOR, the category spaces of the training set and the retrieval set are not the same fine indicating $y_i^f \in \mathcal{Y}^f = \{c_j^f\}_{j=1}^{Y^f}, \hat{y}_i^f \in \hat{\mathcal{Y}}^f = \{\hat{c}_j^f\}_{j=1}^{\hat{Y}^f}$, and $\mathcal{Y}^f \neq \hat{\mathcal{Y}}^f$. $Y^f$ and $\hat{Y}^f$ denote

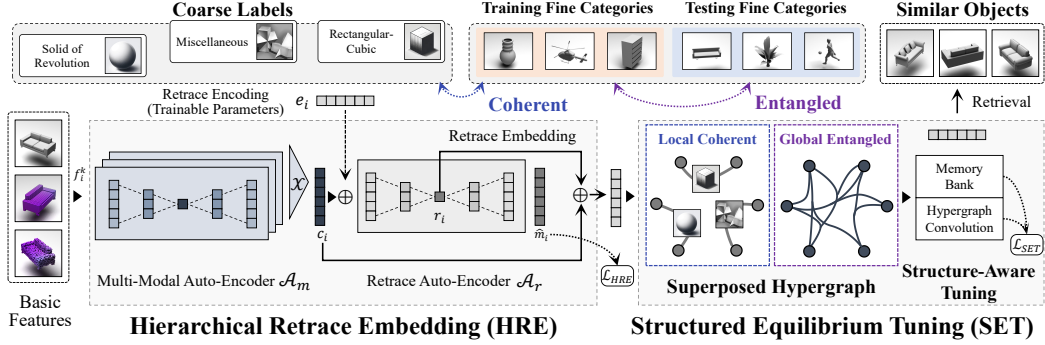

Figure 2: An overview of the proposed Hypergraph-Based Hierarchical Equilibrium Representation framework (HERT) framework for semi-open 3D object retrieval. Our framework is composed of the Hierarchical Retrace Embedding (HRE) and the Structured Equilibrium Tuning (SET) modules, which are designed for multi-level semantic embedding and hierarchical structure-aware tuning.

the numbers of fine categories in the training set and the retrieval set, respectively. In the practice of semi-open 3DOR, the training set and the testing set share the same space of coarse category, which $y_i^c \in \mathcal{Y}^c = \{c_k^c\}_{k=1}^{Y^c}$, $\hat{y}_i^c \in \hat{\mathcal{Y}}^c = \{\hat{c}_k^c\}_{k=1}^{\hat{Y}^c}$, and $\mathcal{Y}^c = \hat{\mathcal{Y}}^c$, $Y^c$ and $\hat{Y}^c$ denote the numbers of coarse categories in the training set and the testing set, respectively. During the retrieval phase, each query object is provided with a coarse category label. For better representation, the semi-open 3DOR defines the retrieval model $r := (o_i|y_i^c) \mapsto z_i$ that maps the 3D object $o_i$ into a semantic embedding $z_i \in \mathbb{R}^d$ under coarse category condition $y_i^c$. The task seeks to minimize the expected risk under the conditional constraints of coarse labels:

$$\mathbb{E}_{(S_i,S_j)\sim(\mathcal{S}_q,\mathcal{S}_t)}\left[\mathbb{I}(\hat{y}_i^f \neq \hat{y}_j^f)e^{-\mathbb{D}(r(o_i|y_i^c),r(o_j|y_j^c))} + \mathbb{I}(\hat{y}_i^f = \hat{y}_j^f)(1 - e^{-\mathbb{D}(r(o_i|y_i^c),r(o_j|y_j^c))})\right], \quad (1)$$

where $S_i = (o_i, y_i^c, \hat{y}_i^f)$ and $S_j = (o_j, y_j^c, \hat{y}_j^f)$ are object instances sampled from the query set $\mathcal{S}_q$ and target set $\mathcal{S}_t$, $\hat{y}_i^f$ and $\hat{y}_j^f$ denote the predicted fine labels, $y_i^c$ and $y_j^c$ are the conditional constraint, which are coarse labels. $\mathbb{I}(\cdot)$ is the indicator function, which returns 1 if the expression holds and 0 otherwise. $\mathcal{H}$ is the hypothesis space of map $r(\cdot|\cdot)$. $\mathbb{D}(z_i, z_j)$ is the distance metric function between different embeddings.

## 4 Methodology

### 4.1 Overall Framework

The overall framework of the Hypergraph-Based Hierarchical Equilibrium Representation (HERT) framework is illustrated in Figure 2. HERT is composed of the *Hierarchical Retrace Embedding (HRE)* and the *Structured Equilibrium Tuning (SET)* modules. Given 3D objects represented by multiple modalities, common-used backbones are used to extract the basic features for each modality. Then, the HRE module is introduced to generate the multi-level retrace embeddings of hierarchical semantic information. Next, in the SET module, the superposed hypergraph is constructed based on the local coherent and global entangled correlations. Finally, the hypergraph convolution and memory bank under the superposed structure are used to smooth and distill for feature generalization between seen and unseen categories.

### 4.2 Hierarchical Retrace Embedding

To obtain fully multi-level embeddings of objects based on hierarchical categories, the HRE module is designed here. Specifically, the HRE utilizes two hierarchical auto-encoders in series as shown in Figure 2. The multi-modal auto-encoder $\mathcal{A}_m$ encodes the multi-modal basic features of 3D objects

to get the unified embeddings. The retrace auto-encoder $\mathcal{A}_r$ encodes the coarse label aligning with the unified embeddings to get retrace embeddings.

Given $N$ 3D objects $\{o_i\}_{i=1}^N$ and feature extractors $\{\mathcal{F}^k\}_{k=1}^M$, the basic feature of $M$ modalities $\{\mathbf{F}^k\}_{k=1}^M = \mathcal{F}^k(\{o_i\}_{i=1}^N)$ can be generated, where $\mathbf{F}^k \in \mathbb{R}^{N \times d_f}$. As shown in Fig. 2, we first compress the input basic features from $M$ modalities into the latent modal-invariant space and generate the unified embeddings from multi-modal auto-encoder $\mathcal{A}_m$. Specifically, for the basic features $f_i^k$ of object $o_i$, the unified embedding of it can be denoted as $c_i = \mathcal{T}\left(\mathcal{A}_m(\{f_i^k\}_{k=1}^M)\right)$, where $\mathcal{X}$ denotes the aggregation function among modalities and $c_i \in \mathbb{R}^{d_c}$.

After getting the unified embedding $c_i$, the retrace auto-encoder $\mathcal{A}_r$ takes the coarse category label $y_i^c$ into retrace encoding $e_i$. As shown in Fig. 2, then $\mathcal{A}_r$ compresses the unified embedding $c_i$ aligned with $e_i$ into the retracte space $\mathbb{S}_r$ and does the reverse reconstruction to the mixed space $\mathbb{S}_x$, which can be defined as follows: $\Psi := \mathbb{S}_x \to \mathbb{S}_f$ and $\Phi := \mathbb{S}_f \to \mathbb{S}_x$, where $\Psi(\cdot)$ is the encoder that maps the unified embedding $c_i$ aligned with $e_i$ into the retracte space $\mathbb{S}_r$, the retrace embedding $r_i$ can be generated by as $r_i = \Psi(c_i + e_i), r_i \in \mathbb{R}^{d_r}$. $\Phi(\cdot)$ is the decoder that maps the retrace embedding to reconstruction feature $\hat{m}_i = \Phi(r_i), \hat{m}_i \in \mathbb{R}^{d_c}$.

Through the HRE stage, we got the unified embedding $c_i$, retrace embedding $r_i$, and mixed feature $\hat{m}_i$ for each 3D object.

## 4.3 Structured Equilibrium Tuning

To endow the unified and retrace embeddings of 3D objects with the ability to generalize to unseen categories, we introduce the SET module as shown in Figure 2. Specifically, the *Superposed Hypergraph* structure is employed to capture the local coherent and global entangled correlations under the constraint of hierarchical category information. Then, hypergraph convolution is employed to utilize the collaborative high-order correlation under the guidance of this superposed Hypergraph. Finally, we use the hypergraph convolution and memory bank to generalize the structure-aware knowledge to generate unbiased features for unseen categories.

### 4.3.1 Superposed Hypergraph

Despite the lack of fine labels in objects from unseen categories, the implicit information within these categories can enhance the generalization capabilities of 3D object embeddings, especially when prompted by coarse labels. To establish the high-order correlations between 3D objects from both seen and unseen categories under common general space, we design a superposed hypergraph structure. The hypergraph in this paper can be represented as $\mathcal{G} = \{\mathcal{V}, \mathcal{E}\}$, where $\mathcal{V}$ and $\mathcal{E}$ are the vertex set and the hyperedge set, respectively.

We construct the vertices in our superposed hypergraph by the multi-level retrace embeddings. Specifically, we combine the unified embedding $c_i$ and retrace embedding $r_i$ to generate the heterogeneous retrace vertices $v_i \in \mathbb{R}^{N \times d_c}$, which can be defined as follows:

$$v_i = \lambda c_i + (1 - \lambda) r_i \tag{2}$$

where $\lambda$ is the hyper-parameter to trade-off between embeddings of different levels.

Distinct from the traditional hypergraph [8], we construct a superposed hypergraph with two types of conditional hyperedges: coherent hyperedges and entangled hyperedges.

**Coherent Hyperedge.** The coherent hyperedges capture the local coherent correlations in the coarse category space. We define the coherent hyperedge as $e^c \in \mathcal{E}_c$ under the condition of coarse labels:

$$\mathcal{E}_c = \{\mathcal{S}_v(y^c) \mid y^c \in \mathcal{Y}^c\}, \tag{3}$$

where $\mathcal{S}_v(y^c)$ denotes the vertex subset that shares the same coarse label $y^c$ and $\mathcal{Y}^c$ denotes the coarse label space of 3D objects.

**Entangled Hyperedge.** The entangled hyperedges model the global collaborative correlations among objects between seen and unseen categories. For each vertex, we define its entangled hyperedge as the set of nearest $K - 1$ neighboring vertices. Specifically,

$$\mathcal{E}_t = \{\mathcal{D}_{\text{KNN}_k}(v) \mid v \in \mathcal{V}\} \tag{4}$$

where $\mathcal{D}_{\mathrm{KNN}_k}(v)$ denotes the k-nearest neighbors in the feature space of vertex $v$.

In this way, we construct $Y^c$ coherent hyperedges and $N$ entangled hyperedges, where $Y^c$ is the number of coarse categories and $N$ is the number of vertex in the superposed hypergraph. Finally, we combine the two sets of hyperedges to obtain a complete superposed hypergraph for structure-aware tuning.

$$\mathcal{E} = \mathcal{E}_c \cup \mathcal{E}_t \tag{5}$$

where $\mathcal{E}_c$ and $\mathcal{E}_t$ are the set of coherent and entangled hyperedges, respectively.

### 4.3.2 Structure-Aware Tuning

To get better generalization on unseen categories, hypergraph convolution and memory bank are adopted here for feature smoothing and distillation.

For the convenience of computation, the hypergraph can be represented by the incidence matrix $\mathbf{H} \in \{0,1\}^{|\mathcal{V}| \times |\mathcal{E}|}$, where the $i$-th hyperedge is the $i$-th column of $\mathbf{H}$, and $\mathbf{H}(v,e) = 1$ if the hyperedge $e$ contains the vertex $v$. $\mathbf{W} \in \mathbb{R}^{|\mathcal{E}| \times |\mathcal{E}|}$ is a diagonal matrix, where $\mathbf{W}_{i,i}$ denotes the weight of the $i$-th hyperedge.

To learn the conditional embeddings $\tilde{\mathbf{V}} \in \mathbb{R}^{N \times d_c}$ from multi-level embedding $\mathbf{V} \in \mathbb{R}^{N \times d_c}$ under the guidance of hypergraph, the hypergraph convolution (HGNNConv [8]) can be represented as:

$$\tilde{\mathbf{V}} = \sigma \left( \mathbf{D}_v^{-\frac{1}{2}} \mathbf{H} \mathbf{W} \mathbf{D}_e^{-1} \mathbf{H}^\top \mathbf{D}_v^{-\frac{1}{2}} \mathbf{V} \boldsymbol{\Theta} \right), \tag{6}$$

where $\mathbf{D}_v$ and $\mathbf{D}_e$ are the diagonal degree matrices for vertex and hyperedge, respectively. $\boldsymbol{\Theta} \in \mathbb{R}^{d_u \times d_u}$ is the trainable parameter for the HGNNConv layer [8].

To increase the generalization ability of the SET module, we construct a memory bank $\mathcal{M}$ that contains $L$ invariant memory anchors for conditional embedding $\tilde{v}_i$ of the 3D object $o_i$, we compute the activation score for each memory anchor in the memory bank by $t_{ij} = \mathcal{D}_m(\tilde{v}_i, a_j)$, where $a_j$ denotes the anchor and $D_m(\cdot, \cdot)$ denotes the distance metric function. We rebuild the aligned multi-level embedding of each object by $z_i = \sum_{j=1}^L t_{i,j}^n a_j, z_i \in \mathbb{R}^{d_c}$, where $t_{ij}^n$ denotes the normalized values of activation score.

### 4.4 Training Objective

**Loss Function for the HRE.** To get a better representation of multi-level label information, we first use the Homology Loss $\mathcal{L}_{homo}$ and Bi-reconstruction Loss $\mathcal{L}_{br}$ followed [7] for $\mathcal{A}_m$ to leverage the collaborative information across modalities, then we adopt the Retrace Cross-Entropy loss $\mathcal{L}_{ce}$ to guide the retrace embedding of coarse category information, which can be defined as follows:

$$\mathcal{L}_{ce} = -\sum_{k=1}^{Y^c} y_{i,k}^c \log(p_{i,k}), \tag{7}$$

where $p_{i,k} = \frac{e^{\hat{m}_{i,k}}}{\sum_{k=1}^{Y^c} e^{\hat{m}_{i,k}}}$ is the predicted probability score of the 3D object $o_i$ in $k$-th coarse category for the reconstructed mixed feature $\hat{m}_i$. $y_{i,k}^c$ is the $k$-th value of the one-hot encoded ground truth coarse label of $o_i$, and $Y^c$ is the number of the coarse categories.

In the hierarchical retrace embedding stage, the overall loss function is given:

$$\mathcal{L}_{HRE} = \mu(\mathcal{L}_{homo} + \mathcal{L}_{br}) + (1-\mu)\mathcal{L}_{ce} \tag{8}$$

where $\mu$ is the hyper-parameter to trade-off between the loss of multi-modal and multi-level representations.

**Loss Function for the SET.** To train the hypergraph convolution and learnable memory anchors, we adopt Memory Reconstruction Loss $\mathcal{L}_{mr}$ and the Cross-entropy Loss $\mathcal{L}_{ce}$:

$$\mathcal{L}_{mr} = \left\| \tilde{u}_i - z_i \right\|_2, \tag{9}$$

$$\mathcal{L}_{ce} = -\sum_{k=1}^{Y} \left( y_{i,k} \log(\tilde{p}_{i,k}) + y_{i,k} \log(p_{i,k}) \right), \tag{10}$$

Table 1: Comparisons of retrieval performance on SO-ESB and SO-NTU dataset.

| Method | SO-ESB | | | | SO-NTU | | | |
|---|---|---|---|---|---|---|---|---|
| | mAP↑ | Recall↑ | NDCG↑ | ANMRR↓ | mAP↑ | Recall↑ | NDCG↑ | ANMRR↓ |
| SDML | 0.4947 | 0.8027 | 0.1858 | 0.5430 | 0.4384 | 0.7009 | 0.1937 | 0.5764 |
| CMCL | 0.4990 | 0.8154 | 0.1880 | 0.5457 | 0.4440 | 0.7053 | 0.1946 | 0.5721 |
| MMSAE | 0.5036 | 0.8503 | 0.1931 | 0.5523 | 0.4454 | 0.7046 | 0.1935 | 0.5745 |
| TranGCN | 0.5063 | 0.9011 | 0.1968 | 0.5408 | 0.4548 | 0.7121 | 0.1961 | 0.5624 |
| C2AE | 0.4809 | 0.7863 | 0.1824 | 0.5501 | 0.4303 | 0.6987 | 0.1915 | 0.5828 |
| HGM$^2$R | 0.5049 | 0.8831 | 0.1939 | 0.5551 | 0.4821 | 0.7364 | 0.2026 | 0.5438 |
| **Ours** | **0.5756** | **0.9346** | **0.2045** | **0.4874** | **0.5678** | **0.8116** | **0.2251** | **0.4677** |

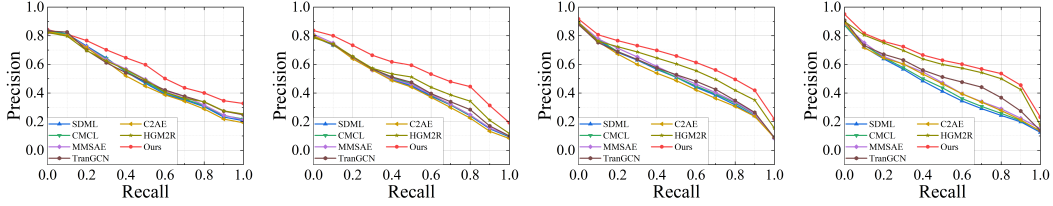

(a) PR-C on SO-ESB.  (b) PR-C on SO-NTU.  (c) PR-C on SO-MN40.  (d) PR-C on SO-ABO.

Figure 3: The Precision-Recall Curves (PR-C) of the proposed method and compared methods on the four datasets, respectively.

where $\tilde{p}_{i,k} = \frac{e^{\tilde{u}_{i,k}}}{\sum_{m=1}^{Y} e^{\tilde{u}_{i,m}}}$ and $p_{i,k} = \frac{e^{z_{i,k}}}{\sum_{m=1}^{Y} e^{z_{i,m}}}$ is the predicted probability score of the 3D object $o_i$ in $k$-th fine category for the multi-level embedding $\tilde{v}_i$ and memory reconstruction embedding $z_i$. $y_{i,k}$ is the $k$-th value of the one-hot encoded ground truth fine label of $o_i$, and $Y$ is the number of the coarse categories.

In the structured equilibrium tuning stage, the overall loss function is given by:

$$\mathcal{L}_{SET} = \eta \mathcal{L}_{mr} + (1 - \eta) \mathcal{L}_{ce} \tag{11}$$

where $\eta$ is the hyper-parameter for trade-off between them.

## 5 Experiments

### 5.1 Dataseta and Evaluation Metrics

**Datasets.** We generate four semi-open 3DOR datasets, including SO-ESB, SO-NTU, SO-MN40, and SO-ABO, based on the public datasets ESB [13], NTU [5], ModelNet40 [35], and ABO [6], respectively. We add coarse category labels for each object based on the basic geometric shapes such as solid of revolution, rectangular-cubic, *etc*. Also, we remove some objects that are difficult to categorize based on their shapes. Then, we split the fine categories into seen categories for training and unseen categories for testing, the training and testing sets share the same coarse label space according to the semi-open environment setting. Each object has three modalities including multi-view, voxel, and point cloud. Specifically, the detailed descriptions of dataset generation and setting are shown in Appendix B.

**Evaluation Protocols.** As for the evaluation criteria, We employ commonly used retrieval metrics for comparison, including Mean Average Precision (mAP), Recall, Normalized Discounted Cumulative Gain (NDCG), Average Normalized Modified Retrieval Rank (ANMRR), and the Precision-Recall Curve (PR-C). For the mAP, Recall, and NDCG metrics, the higher scores is better. For the ANMRR metric, the lower score is better.

Table 2: Comparisons of retrieval performance on SO-MN40 and SO-ABO dataset.

| Method | SO-MN40 | | | | SO-ABO | | | |
|---|---|---|---|---|---|---|---|---|
| | mAP↑ | Recall↑ | NDCG↑ | ANMRR↓ | mAP↑ | Recall↑ | NDCG↑ | ANMRR↓ |
| SDML | 0.5018 | 0.3241 | 0.6082 | 0.5106 | 0.4380 | 0.3425 | 0.4726 | 0.5564 |
| CMCL | 0.5086 | 0.3281 | 0.6128 | 0.5060 | 0.4520 | 0.3657 | 0.4816 | 0.5458 |
| MMSAE | 0.5189 | 0.3335 | 0.6226 | 0.4938 | 0.4783 | 0.3863 | 0.4929 | 0.5264 |
| TranGCN | 0.5188 | 0.3358 | 0.6131 | 0.4957 | 0.5175 | 0.3956 | 0.5127 | 0.4801 |
| C2AE | 0.4865 | 0.3152 | 0.5977 | 0.5231 | 0.4669 | 0.3674 | 0.4794 | 0.5313 |
| HGM$^2$R | 0.5779 | 0.3698 | 0.6482 | 0.4407 | 0.6069 | 0.4675 | 0.5463 | 0.4154 |
| **Ours** | **0.6336** | **0.3993** | **0.6874** | **0.3972** | **0.6339** | **0.4793** | **0.5622** | **0.3836** |

Table 3: Ablation studies on SO-ESB and SO-NTU dataset.

| Method | SO-ESB | | | | SO-NTU | | | |
|---|---|---|---|---|---|---|---|---|
| | mAP↑ | Recall↑ | NDCG↑ | ANMRR↓ | mAP↑ | Recall↑ | NDCG↑ | ANMRR↓ |
| HRE w/o ReEnz | 0.5159 | 0.9086 | 0.1953 | 0.5431 | 0.4913 | 0.7534 | 0.2053 | 0.5355 |
| HRE w/o $\mathcal{L}_{ce}$ | 0.5133 | 0.8738 | 0.1934 | 0.5365 | 0.5161 | 0.7902 | 0.2162 | 0.5162 |
| SET w/o $\mathcal{E}_c$ | 0.5358 | 0.8957 | 0.1975 | 0.5184 | 0.5285 | 0.7898 | 0.2184 | 0.4986 |
| GCN-based SET | 0.5405 | 0.8999 | 0.2003 | 0.5192 | 0.5144 | 0.7703 | 0.2140 | 0.5138 |
| MLP-based SET | 0.5014 | 0.8483 | 0.1930 | 0.5476 | 0.4689 | 0.7304 | 0.2023 | 0.5561 |
| **HRE+SET** | **0.5756** | **0.9346** | **0.2045** | **0.4874** | **0.5678** | **0.8116** | **0.2251** | **0.4677** |

## 5.2 Experimental Settings

**Implemental Details.** In our experiments, we choose three modes of multi-view (12 views), point cloud (1024 points), and voxel (32 dimensions) as the representation of 3D objects. The basic features for framework input are extracted by MVCNN [30], PointNet [25], and 3D ShapeNets [35], respectively. The HRE and SET modules are trained separately with 40 and 120 epochs. The SGD optimizers are used for both two modules with learning rates of 0.1 and 0.001, respectively. As for the hyper-parameters in HERT, we set $\lambda = 0.5$, $\mu = 0.8$, and $\eta = 0.9$. Detailed implemental settings for our framework are provided in Appendix C.

**Compared Methods.** Under this semi-open setting, since there is no 3D object retrieval method designed specifically for this muli-level settings, we choose the current state-of-the-art methods of close-set 3D retrieval (SDML [11], CMCL [14], MMSAE [34]), close-set multi-label retrieval(TranGCN [15]), and open-set 3D learning (C2AE [21], HGM$^2$R [7]). For each method, we add a multi-label learning mechanism [15] on their basis for comparison. We provide more implemented details of compared methods in Appendix D.

## 5.3 Comparison with the State-of-the-Arts

To evaluate the effectiveness of the proposed HERT framework, we conduct experiments on SO-ESB, SO-NTU, SO-MN40, and SO-ABO datasets. The comparison of quantitative results is presented in Table 1 and Table 2, respectively. Results demonstrate that the proposed HERT outperforms the state-of-the-art methods on all four datasets. Especially on the SO-NTU and SO-MN40 datasets, our method achieves mAP of $0.5678/0.6336$ with about $17.7\%/9.6\%$ improvements compared with the second-best method, and achieves Recall of $0.8116/0.3993$ with about $10.2\%/7.9\%$ improvements compared with the second-best method. Besides, we also provide the qualitative results through precision-recall curves as shown in Figure 3, in which the larger area below the curve indicates better performance.

The better results indicate that the proposed HERT framework has the capability to understand and generalize unseen fine categories under the guidance of coarse labels. The proposed HRE and SET modules can fully leverage the hierarchical category information into multi-level retrace embedding and generalize them to unseen categories. This approach better captures the hierarchical semantic correlations in the wild and provides a practical framework for the representation learning of multi-label tasks in semi-open environments. We provide more visualized results in Appendix E.

Table 4: Ablation studies on SO-MN40 and SO-ABO dataset.

| Method | SO-MN40 | | | | SO-ABO | | | |
|---|---|---|---|---|---|---|---|---|
| | mAP↑ | Recall↑ | NDCG↑ | ANMRR↓ | mAP↑ | Recall↑ | NDCG↑ | ANMRR↓ |
| HRE w/o ReEnz | 0.5791 | 0.3710 | 0.6479 | 0.4410 | 0.6055 | 0.4523 | 0.5535 | 0.4062 |
| HRE w/o $\mathcal{L}_{ce}$ | 0.5967 | 0.3783 | 0.6756 | 0.4309 | 0.5885 | 0.4269 | 0.5413 | 0.4230 |
| SET w/o $\mathcal{E}_c$ | 0.5913 | 0.3757 | 0.6669 | 0.4347 | 0.6006 | 0.4263 | 0.5494 | 0.4132 |
| GCN-based SET | 0.5602 | 0.3573 | 0.6410 | 0.4628 | 0.5686 | 0.4253 | 0.5314 | 0.4415 |
| MLP-based SET | 0.5088 | 0.3290 | 0.6149 | 0.5073 | 0.4880 | 0.3787 | 0.5023 | 0.5159 |
| **HRE+SET** | **0.6336** | **0.3993** | **0.6874** | **0.3972** | **0.6339** | **0.4793** | **0.5622** | **0.3836** |

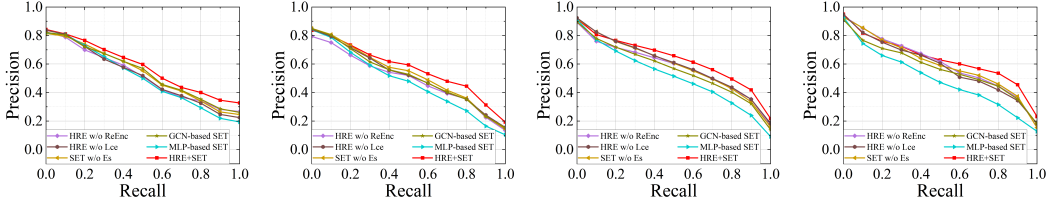

(a) PR-C on SO-ESB.    (b) PR-C on SO-NTU.    (c) PR-C on SO-MN40.    (d) PR-C on SO-ABO.

Figure 4: The Precision-Recall Curves (PR-C) of the ablation studies on four datasets, respectively.

## 5.4 Ablation Study

We conduct ablation experiments on each module of HERT to demonstrate their effectiveness. First, we remove the retrace encoding $e_i$ (HRE w/o ReEnz) and cross-entropy loss in the HRE module (HRE w/o $\mathcal{L}_{ce}$), where "w/o" denotes "without". This is equivalent to using a naive hierarchical embedding approach for coarse labels. As shown in Table 3, 4, and Figure 4, the proposed retrace embedding approach in the HRE module achieves an mAP improvement of $9.4\%/6.2\%$ in SO-MN40 dataset and $7.8\%/4.7\%$ in SO-ABO dataset. These results effectively demonstrate the effectiveness of the HRE module for hierarchical categories.

As for the SET module, we remove the proposed coherent hyperedges (SET w/o $\mathcal{E}_c$) for comparison, also we replace the hypergraph convolution with Graph Convolutional Networks (GCN-based SET) and Multilayer Perceptron (MLP-based SET). Ablation results in Table 3, 4, and Figure 4 show that the proposed SET outperforms all other structure learning methods, and the combination of the HRE and the SET yields the best performance. These results demonstrate the proposed SET can effectively utilize the semi-superposed correlations among objects.

## 6 Conclusion

In this paper, we introduce a more practical *Semi-Open Environment* setting for open-set 3D object retrieval with hierarchical labels, in which the training and testing set share a partial label space for coarse categories but are completely disjoint from fine categories. We propose the Hypergraph-Based Hierarchical Equilibrium Representation (HERT) framework for semi-open 3D object retrieval. Specifically, to overcome the global disequilibrium of unseen categories, we propose the Hierarchical Retrace Embedding (HRE) module to fully leverage the multi-level category information. Besides, we perform the Structured Equilibrium Tuning (SET) module to tackle the feature overlap and class confusion problem. This module utilizes more equilibrial correlations among objects and generalizes to unseen categories, by constructing a superposed hypergraph based on the local coherent and global entangled correlations. Furthermore, we construct four 3D object datasets with multi-level category labels for semi-open 3DOR tasks, *i.e.*, SO-ESB, SO-NTU, SO-MN40, and SO-ABO. Results demonstrate that the proposed method can effectively generate and generalize the hierarchical embeddings of 3D objects in semi-open environments. However, due to dataset limitations, we are currently unable to verify the balanced representation effect on more than three levels of labels, which is one of our future research directions. We believe this paper can provide new insights for future research in more practical scenarios of open-set learning.

# 7 Acknowledgement

This work was supported by Beijing Natural Science Foundation (No. L242167), CCF-Tencent Open Research Fund, and Jiangxi Provincial Natural Science Foundation (20224ACB218002).

## Footnotes

[2]`https://www.blender.org`

[3]`https://www.open3d.org`

# References

[1] Ferran Argelaguet and Carlos Andujar. A survey of 3d object selection techniques for virtual environments. *Computers & Graphics*, 37(3):121–136, 2013.

[2] Eduardo Arnold, Omar Y Al-Jarrah, Mehrdad Dianati, Saber Fallah, David Oxtoby, and Alex Mouzakitis. A survey on 3d object detection methods for autonomous driving applications. *IEEE Transactions on Intelligent Transportation Systems*, 20(10):3782–3795, 2019.

[3] Song Bai, Peng Tang, Philip HS Torr, and Longin Jan Latecki. Re-ranking via metric fusion for object retrieval and person re-identification. In *IEEE/CVF Conference on Computer Vision and Pattern Recognition*, pages 740–749, 2019.

[4] Abhijit Bendale and Terrance E Boult. Towards open set deep networks. In *IEEE/CVF Conference on Computer Vision and Pattern Recognition*, pages 1563–1572, 2016.

[5] Ding-Yun Chen, Xiao-Pei Tian, Yu-Te Shen, and Ming Ouhyoung. On visual similarity based 3d model retrieval. *Computer graphics forum*, 22(3):223–232, 2003.

[6] Jasmine Collins, Shubham Goel, Kenan Deng, Achleshwar Luthra, Leon Xu, Erhan Gundogdu, Xi Zhang, Tomas F Yago Vicente, Thomas Dideriksen, Himanshu Arora, et al. Abo: Dataset and benchmarks for real-world 3d object understanding. In *IEEE/CVF Conference on Computer Vision and Pattern Recognition*, pages 21126–21136, 2022.

[7] Yifan Feng, Shuyi Ji, Yu-Shen Liu, Shaoyi Du, Qionghai Dai, and Yue Gao. Hypergraph-based multi-modal representation for open-set 3d object retrieval. *IEEE Transactions on Pattern Analysis and Machine Intelligence*, 46(4):2206–2223, 2023.

[8] Yue Gao, Yifan Feng, Shuyi Ji, and Rongrong Ji. Hgnn+: General hypergraph neural networks. *IEEE Transactions on Pattern Analysis and Machine Intelligence*, 45(3):3181–3199, 2022.

[9] ZongYuan Ge, Sergey Demyanov, Zetao Chen, and Rahil Garnavi. Generative openmax for multi-class open set classification. *arXiv preprint arXiv:1707.07418*, 2017.

[10] Alexander Grabner, Peter M Roth, and Vincent Lepetit. 3d pose estimation and 3d model retrieval for objects in the wild. In *IEEE/CVF Conference on Computer Vision and Pattern Recognition*, pages 3022–3031, 2018.

[11] Peng Hu, Liangli Zhen, Dezhong Peng, and Pei Liu. Scalable deep multimodal learning for cross-modal retrieval. In *Annual Conference of the Association for Computing Machinery Special Interest Group in Information Retrieval*, pages 635–644, 2019.

[12] Lalit P Jain, Walter J Scheirer, and Terrance E Boult. Multi-class open set recognition using probability of inclusion. In *European Conference on Computer Vision*, pages 393–409. Springer, 2014.

[13] Subramaniam Jayanti, Yagnanarayanan Kalyanaraman, Natraj Iyer, and Karthik Ramani. Developing an engineering shape benchmark for cad models. *Computer-Aided Design*, 38(9):939–953, 2006.

[14] Longlong Jing, Elahe Vahdani, Jiaxing Tan, and Yingli Tian. Cross-modal center loss for 3d cross-modal retrieval. In *IEEE/CVF Conference on Computer Vision and Pattern Recognition*, pages 3142–3151, 2021.

[15] Ying Li, Chunming Guan, Rui Cai, Ye Erwan, Ding Yuxiang, and Jiaquan Gao. Tran-gcn: Multi-label pattern image retrieval via transformer driven graph convolutional network. In *ACM International Conference on Multimedia*, pages 6301–6310, 2023.

[16] Ying Li, Hongwei Zhou, Yeyu Yin, and Jiaquan Gao. Multi-label pattern image retrieval via attention mechanism driven graph convolutional network. In *ACM International Conference on Multimedia*, pages 300–308, 2021.

[17] Qi Liang, Mengmeng Xiao, and Dan Song. 3d shape recognition based on multi-modal information fusion. *Multimedia Tools and Applications*, 80:16173–16184, 2021.

[18] Weiwei Liu, Donna Xu, Ivor W Tsang, and Wenjie Zhang. Metric learning for multi-output tasks. *IEEE Transactions on Pattern Analysis and Machine Intelligence*, 41(2):408–422, 2018.

[19] Lawrence Neal, Matthew Olson, Xiaoli Fern, Weng-Keen Wong, and Fuxin Li. Open set learning with counterfactual images. In *European Conference on Computer Vision*, pages 613–628, 2018.

[20] Weizhi Nie, Qi Liang, An-An Liu, Zhendong Mao, and Yangyang Li. Mmjn: Multi-modal joint networks for 3d shape recognition. In *ACM International Conference on Multimedia*, pages 908–916, 2019.

[21] Poojan Oza and Vishal M Patel. C2ae: Class conditioned auto-encoder for open-set recognition. In *IEEE/CVF Conference on Computer Vision and Pattern Recognition*, pages 2307–2316, 2019.

[22] Panagiotis Papadakis, Ioannis Pratikakis, Theoharis Theoharis, and Stavros Perantonis. Panorama: A 3d shape descriptor based on panoramic views for unsupervised 3d object retrieval. *International Journal of Computer Vision*, 89:177–192, 2010.

[23] Jitendra Parmar, Satyendra Chouhan, Vaskar Raychoudhury, and Santosh Rathore. Open-world machine learning: applications, challenges, and opportunities. *ACM Computing Surveys*, 55(10):1–37, 2023.

[24] Yashoteja Prabhu, Anil Kag, Shrutendra Harsola, Rahul Agrawal, and Manik Varma. Parabel: Partitioned label trees for extreme classification with application to dynamic search advertising. In *The International Conference of World Wide Web*, pages 993–1002, 2018.

[25] Charles R Qi, Hao Su, Kaichun Mo, and Leonidas J Guibas. Pointnet: Deep learning on point sets for 3d classification and segmentation. In *IEEE/CVF Conference on Computer Vision and Pattern Recognition*, pages 652–660, 2017.

[26] Ethan M Rudd, Lalit P Jain, Walter J Scheirer, and Terrance E Boult. The extreme value machine. *IEEE Transactions on Pattern Analysis and Machine Intelligence*, 40(3):762–768, 2017.

[27] Walter J Scheirer, Anderson de Rezende Rocha, Archana Sapkota, and Terrance E Boult. Toward open set recognition. *IEEE Transactions on Pattern Analysis and Machine Intelligence*, 35(7):1757–1772, 2012.

[28] Walter J Scheirer, Lalit P Jain, and Terrance E Boult. Probability models for open set recognition. *IEEE Transactions on Pattern Analysis and Machine Intelligence*, 36(11):2317–2324, 2014.

[29] Lei Shu, Hu Xu, and Bing Liu. Doc: Deep open classification of text documents. *arXiv preprint arXiv:1709.08716*, 2017.

[30] Hang Su, Subhransu Maji, Evangelos Kalogerakis, and Erik Learned-Miller. Multi-view convolutional neural networks for 3d shape recognition. In *IEEE/CVF Conference on Computer Vision and Pattern Recognition*, pages 945–953, 2015.

[31] Jian Wang, Fan Li, Xuchong Zhang, and Hongbin Sun. Adversarial obstacle generation against lidar-based 3d object detection. *IEEE Transactions on Multimedia*, 2023.

[32] Shijie Wang, Jianlong Chang, Haojie Li, Zhihui Wang, Wanli Ouyang, and Qi Tian. Open-set fine-grained retrieval via prompting vision-language evaluator. In *IEEE/CVF Conference on Computer Vision and Pattern Recognition*, pages 19381–19391, 2023.

[33] Dayan Wu, Zheng Lin, Bo Li, Mingzhen Ye, and Weiping Wang. Deep supervised hashing for multi-label and large-scale image retrieval. In *International Conference on Multimedia Retrieval*, pages 150–158, 2017.

[34] Yiling Wu, Shuhui Wang, and Qingming Huang. Multi-modal semantic autoencoder for cross-modal retrieval. *Neurocomputing*, 331:165–175, 2019.

[35] Zhirong Wu, Shuran Song, Aditya Khosla, Fisher Yu, Linguang Zhang, Xiaoou Tang, and Jianxiong Xiao. 3d shapenets: A deep representation for volumetric shapes. In *IEEE/CVF Conference on Computer Vision and Pattern Recognition*, pages 1912–1920, 2015.

[36] Zhi Xiong, Dayan Wu, Wen Gu, Haisu Zhang, Bo Li, and Weiping Wang. Deep discrete attention guided hashing for face image retrieval. In *International Conference on Multimedia Retrieval*, pages 136–144, 2020.

[37] Yang Xu, Yifan Feng, and Lin Bie. Triadic elastic structure representation for open-set incremental 3d object retrieval. In *ACM International Conference on Multimedia Retrieval*, pages 20–28, 2024.

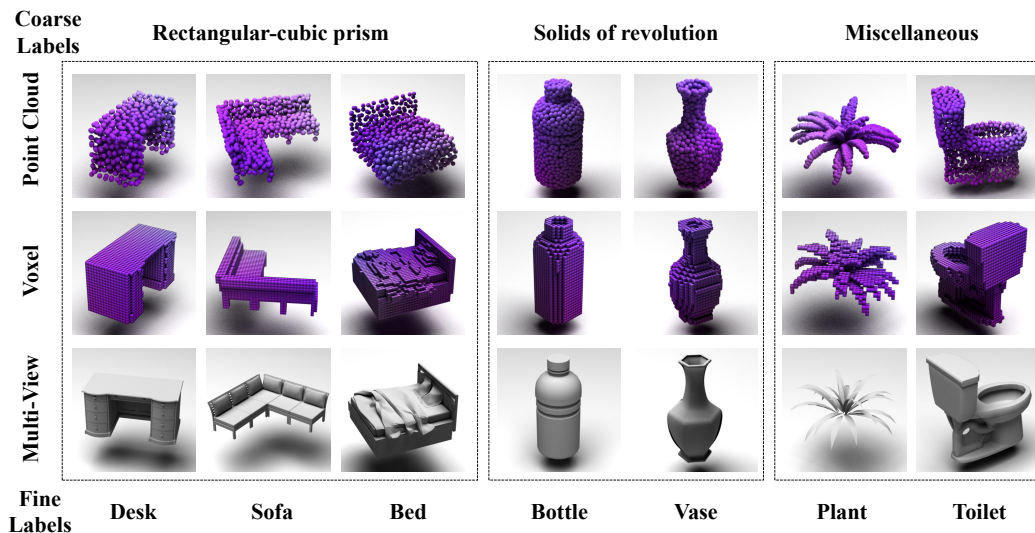

Figure 5: Visualizations of the hierarchical labels and multi-modal representations of 3D objects in the SO-MN40 datasets.

[38] Yang Xu, Yifan Feng, and Yue Gao. Negative prompt driven complementary parallel representation for open-world 3d object retrieval. In *International Joint Conference on Artificial Intelligence*, pages 1498–1506, 2024.

[39] Yang Xu, Yifan Feng, and Yu Jiang. Structure-aware residual-center representation for self-supervised open-set 3d cross-modal retrieval. In *IEEE International Conference on Multimedia and Expo*, pages 1–6, 2024.

[40] Ian EH Yen, Xiangru Huang, Wei Dai, Pradeep Ravikumar, Inderjit Dhillon, and Eric Xing. Ppdsparse: A parallel primal-dual sparse method for extreme classification. In *ACM SIGKDD Conference on Knowledge Discovery and Data Mining*, pages 545–553, 2017.

[41] Ryota Yoshihashi, Wen Shao, Rei Kawakami, Shaodi You, Makoto Iida, and Takeshi Naemura. Classification-reconstruction learning for open-set recognition. In *IEEE/CVF Conference on Computer Vision and Pattern Recognition*, pages 4016–4025, 2019.

[42] Haoxuan You, Yifan Feng, Rongrong Ji, and Yue Gao. Pvnet: A joint convolutional network of point cloud and multi-view for 3d shape recognition. In *ACM International Conference on Multimedia*, pages 1310–1318, 2018.

[43] Haoxuan You, Yifan Feng, Xibin Zhao, Changqing Zou, Rongrong Ji, and Yue Gao. Pvrnet: Point-view relation neural network for 3d shape recognition. In *AAAI Conference on Artificial Intelligence*, pages 9119–9126, 2019.

[44] Hsiang-Fu Yu, Prateek Jain, Purushottam Kar, and Inderjit Dhillon. Large-scale multi-label learning with missing labels. In *International Conference on Machine Learning*, pages 593–601. PMLR, 2014.

[45] Da-Wei Zhou, Han-Jia Ye, and De-Chuan Zhan. Learning placeholders for open-set recognition. In *IEEE/CVF Conference on Computer Vision and Pattern Recognition*, pages 4401–4410, 2021.

[46] Zhi-Hua Zhou. Open-environment machine learning. *National Science Review*, 9(8):nwac123, 2022.

## A    Semi-Open Environment

In practical scenarios, the categories of 3D objects can be labeled from various perspectives such as appearance and functionality, or from different levels of granularity, which means hierarchical labels. For example, consider a 3D object, which can simultaneously be labeled as "sphere"and "blue"at the basic geometric and color level. Moving up the semantic hierarchy, the same object may be labeled as "transportation", "car", and "red"to indicate

Table 5: The statistics of the semi-open 3DOR datasets.

| | | SO-ESB | SO-NTU | SO-MN40 | SO-ABO |
|---|---|---|---|---|---|
| Categories | Coarse | 3 | 3 | 3 | 3 |
| | Fine | 41 | 67 | 40 | 21 |
| | Seen | 17 | 13 | 8 | 4 |
| | Unseen | 24 | 54 | 32 | 17 |
| Number | Training | 98 | 378 | 2821 | 1082 |
| | Retrieval | 457 | 1232 | 7591 | 4432 |
| | Query | 96 | 216 | 128 | 68 |
| | Target | 361 | 1016 | 7463 | 4364 |

its coarse semantic, fine semantic, and color. Further up the hierarchy, labels could include descriptors related to the object's specific function or detailed features. For instance, a 3D architectural model might be labeled as "residential", "multi-story building", and "with a garage"to convey its functional and structural attributes. These hierarchical labels not only enhance the descriptive richness of 3D object representations but also play a crucial role in applications spanning computer graphics, virtual reality, machine learning, and game development. They enable a more comprehensive understanding of an object's characteristics and context, ultimately enhancing its usability and interpretability across diverse domains.

Most open-set learning methods are based on the assumption that the labels of object categories are at a single level. This setting implies that existing methods consider category visibility only between the training and testing sets at a single layer, ignoring the distribution overlap among categories at other layers. Besides, the existing strict setting of the single-level open-set also brings more deviation bias to the feature space of unseen categories. In this paper, we do not follow the strict assumption of the existing open-set approach regarding the disjoint distribution between training and testing sets, instead leveraging the different overlaps across multi-level categories to enhance representation and retrieval performance. we term this scenario, where the training and testing set share a partial label space of coarse categories but are completely disjoint from fine categories, as *Semi-Open Environment*.

## B   Semi-Open Dataset Generation

Considering that there is no existing dataset for semi-open 3DOR task, we generate four semi-open 3DOR datasets, including SO-ESB, SO-NTU, SO-MN40, and SO-ABO, based on the public datasets ESB [13], NTU [5], ModelNet40 [35], and ABO [6], respectively. We add coarse category labels for each object according to the basic geometric shape and remove some objects that are difficult to categorize based on their shapes. Thus, the number of 3D objects in the released datasets is smaller than that of the original datasets. Specifically, the coarse labels for the four datasets are:

- **SO-ESB**: Flat-thin wall components, Rectangular-cubic prism, Solids of revolution.
- **SO-NTU**: Rectangular-cubic prism, Solids of revolution, Miscellaneous shape.
- **SO-MN40**: Rectangular-cubic prism, Solids of revolution, Miscellaneous shape.
- **SO-ABO**: Rectangular-cubic prism, Solids of revolution, Miscellaneous shape.

Then, we split the fine categories into seen categories for training and unseen categories for testing, the training and testing sets share the same coarse label space according to the semi-open environment setting. The statics of the four semi-open 3DOR datasets are shown in Table 5. Each object has three modalities including multi-view, voxel, and point cloud. The multi-view data are rendered by *Blender 3.0*[2] to get 12 images for each object. The point number for point cloud is 1024, and the dimension for voxel data is $32 \times 32 \times 32$. Point cloud and voxel data are sampled by *Open3D 0.13.0*[3]. The illustration of hierarchical labels and multiple modalities of the semi-open 3DOR datasets are shown in Figure 5.

## C   Method Details

The overall framework of HERT is composed of the *Hierarchical Retrace Embedding (HRE)* and the *Conditional Structure Learning (SET)* modules. Given 3D objects represented by multiple modalities, common-used

**Algorithm 1** Training the HRE module
___
**Input**: Basic features $\{r^k\}_{k=1}^M$ and their coarse labels $\{y_i^c\}_{i=1}^N$ of $N$ instances $\{o_i\}_{i=1}^N$.
**Parameter**: $\mu = 0.5$.
**Output 1**: Unified embeddings $\{c_i\}_{i=1}^N$.
**Output 2**: Retrace embeddings $\{f_i\}_{i=1}^N$.

 1: Let $epoch = 0$;
 2: Initialize multi-modal auto-encoder $\mathcal{A}_m$;
 3: Initialize aggregation function $\mathcal{T}$;
 4: Initialize retrace auto-encoder $\mathcal{A}_r = \{\Psi, \Phi\}$;
 5: Encoding coarse labels as retrace encoding $\{e_i\}_{i=1}^N$;
 6: **while** $epoch \leq 40$ **do**
 7:     Get unified embedding for each modality $\{c_i^1, c_i^2, c_i^3\}_{i=1}^N = \mathcal{A}_m(\{f_i^1, f_i^2, f_i^3\}_{i=1}^N)$.
 8:     Get unified embedding for each object $\{c_i\}_{i=1}^N = \mathcal{T}(\{c_i^1, c_i^2, c_i^3\}_{i=1}^N)$.
 9:     Calculate the Homology Loss $\mathcal{L}_{homo} = \mathcal{L}_{homo}(\{c_i\}_{i=1}^N)$.
10:     Calculate the Bi-reconstruction Loss $\mathcal{L}_{br} = \mathcal{L}_{br}(\{c_i\}_{i=1}^N)$.
11:     Get retrace embedding of each object $\{r_i\}_{i=1}^N = \Psi(\{c_i + e_i\}_{i=1}^N)$.
12:     Get mixed feature of each object $\{\hat{m}_i\}_{i=1}^N = \Phi(\{f_i\}_{i=1}^N)$.
13:     Calculate the Retrace Cross-Entropy Loss $\mathcal{L}_{ce} = \mathcal{L}_{ce}(\{\hat{m}_i\}_{i=1}^N)$.
14:     Calculate loss for the HRE module $\mathcal{L}_{HRE} = \mu(\mathcal{L}_{homo} + \mathcal{L}_{br}) + (1 - \mu)\mathcal{L}_{ce}$.
15:     **if** $\mathcal{L}_{HRE}$ does not converges **then**
16:         Update parameters of $\mathcal{A}_m$, $\mathcal{T}$, and $\mathcal{A}_r$ by $\mathcal{L}_{HRE}$.
17:         $epoch+ = 1$
18:     **else**
19:         Break.
20:     **end if**
21: **end while**
22: **return** Unified embeddings $\{c_i\}_{i=1}^N$, and retrace embeddings $\{r_i\}_{i=1}^N$
___

backbones are used to extract the basic features for each modality. Specifically, the basic features of multi-view, point cloud, and voxel are extracted by the pre-trained MVCNN [30], PointNet [25], and 3D ShapeNets [35], respectively.

The implemental details of the HRE module are provided in Algorithm 1, the HRE utilizes two hierarchical auto-encoders in series. The multi-modal auto-encoder $\mathcal{A}_m$ encodes the multi-modal basic feature of 3D objects to get the unified embeddings. The retrace auto-encoder $\mathcal{A}_a$ encodes the coarse label aligning with the unified embeddings to get retrace embeddings.

As shown in Algorithm 2, we provide the implemental details of the SET module, the *superposed hypergraph* structure $\mathcal{G}$ is employed to capture the local coherent and global entangled correlations under the constraint of hierarchical category information. Then, hypergraph convolution is employed to utilize the collaborative high-order correlation under the guidance of this superposed hypergraph. Finally, we use the hypergraph convolution (HGN-NConv) and memory bank $\mathcal{M}$ to generalize the structure-aware knowledge to generate unbiased features for unseen categories.

Table 6: The hyper-parameters of the HERT framework.

|  | **HRE** | **SET** |
|---|---|---|
| Optimizer | SGD | SGD |
| Learning Rate | 0.1 | 0.001 |
| Momentum | 0.9 | 0.9 |
| Weight Decay | 0.1 | 0 |
| LR Scheduler | Cosine Annealing | Cosine Annealing |
| $T_{max}$ | 40 | 60 |
| $eta_{min}$ | 0.00001 | 0.00001 |
| Max Epochs | 40 | 120 |

Our experiments were conducted on a Tesla V100-32G GPU and an Intel(R) Xeon(R) Silver 4210 CPU @ 2.20GHz. Besides, We provide more hyper-parameters of HERT in Table 6. The hyper-parameters "k" in the SET module are set to 12, 10, 50, and 50 for SO-ESB, SO-NTU, SO-MN40, and SO-ABO, respectively.

---

**Algorithm 2** Training the SET module

---

**Input**: Unified embeddings $\{c_i\}_{i=1}^{N}$, retrace embeddings $\{f_i\}_{i=1}^{N}$, and coarse labels $\{y_i^c\}_{i=1}^{N}$ of $N$ instances $\{o_i\}_{i=1}^{N}$.
**Parameter**: $\lambda = 0.8$, $\eta = 0.9$.
**Output**: Final embeddings $\{\hat{v}_i\}_{i=1}^{N}$.

1: Let $epoch = 0$;
2: Initialize superposed hypergraph $\mathcal{G} = \{\mathcal{V}, \mathbf{W}, \mathcal{E}\}$;
3: Construct vertices $\mathcal{V} = \{v_i\}_{i=1}^{N} = \{\lambda c_i + (1 - \lambda)f_i\}_{i=1}^{N}$;
4: Construct coherent hyperedge $\mathcal{E}_c = \{\mathcal{S}_v(y^c) \mid y^c \in \mathcal{Y}^c\}$;
5: Construct entangled hyperedge $\mathcal{E}_t = \{\mathcal{D}_{\text{KNN}_k}(v) \mid v \in \mathcal{V}\}$;
6: Construct hyperedges $\mathcal{E} = \mathcal{E}_c \cup \mathcal{E}_t$;
7: Initialize weight diagonal matrix $\mathbf{W}$ of $\mathcal{G}$;
8: Calculate diagonal degree matrices $\mathbf{D}_v$ and $\mathbf{D}_e$;
9: Calculate incidence matrix $\mathbf{H}$ of $\mathcal{G}$;
10: Initialize HGNNConv parameters $\mathbf{\Theta}$ of $\mathcal{G}$;
11: Construct memory bank $\mathcal{M}$;
12: **while** $epoch \leq 120$ **do**
13:     Get $\{\tilde{v}_i\}_{i=1}^{N} = \sigma(\mathbf{D}_v^{-\frac{1}{2}}\mathbf{HWD}_e^{-1}\mathbf{H}^{\top}\mathbf{D}_v^{-\frac{1}{2}}\{v_i\}_{i=1}^{N}\mathbf{\Theta})$;
14:     Calculate the Cross-Entropy Loss $\mathcal{L}_{ce} = \mathcal{L}_{ce}(\{\tilde{v}_i\}_{i=1}^{N})$;
15:     Rebuild $\{\tilde{v}_i\}_{i=1}^{N}$ to get $\{z_i\}_{i=1}^{N}$ by $\mathcal{M}$;
16:     Calculate the Momery Reconstruction Loss $\mathcal{L}_{mr} = \mathcal{L}_{mr}(\{\tilde{v}_i\}_{i=1}^{N}, \{z_i\}_{i=1}^{N})$;
17:     Calculate loss for the SET module $\mathcal{L}_{SET} = \eta\mathcal{L}_{mr} + (1 - \eta)\mathcal{L}_{ce}$.
18:     **if** $\mathcal{L}_{SET}$ does not converges **then**
19:         Update parameters of $\mathbf{\Theta}$ and $\mathcal{M}$ by $\mathcal{L}_{SET}$.
20:         $epoch += 1$
21:     **else**
22:         Break.
23:     **end if**
24: **end while**
25: **return** Final embeddings $\{\tilde{v}_i\}_{i=1}^{N}$.
26: Retrieval by $\{\tilde{v}_i\}_{i=1}^{N}$.

---

# D    Comparison Implement

Under this semi-open setting, since there is no 3D object retrieval method designed specifically for this muli-level settings, we choose the current state-of-the-art methods of close-set 3D retrieval (SDML [11], CMCL [14], MM-SAE [34]), close-set multi-label retrieval(Tran-GCN [15]), and open-set 3D learning (C2AE [21], HGM$^2$R [7]). For each method, we add a multi-label learning mechanism [15] on their basis for comparison. The coarse embeddings are supervised by the coarse labels by cross-entropy loss.

- SDML [11]: SDML is a metric learning based method for multi-modal retrieval, which learns projection functions for different modalities independently. We implement the multi-label embedding by adding an auto-encoder after the encoder of the DSAE module.

- CMCL [14]: CMCL is a typical 3DOR network based on cross-modal center loss, which is designed to eliminate cross-modal discrepancy. We construct an auto-encoder and take the center representation of each object to generate the coarse embeddings.

- MMSAE [34]: MMSAE is an auto-encoder based method that compresses features from different modalities into a unified space. We take the semantic code vector of the MMSAE network as the input and generate coarse embeddings by another auto-encoder.

- TranGCN [15]: TranGCN is a multi-label retrieval method based on a graph convolutional network. We treat the fine and coarse labels as the two labels of supervision.

- C2AE [21]: C2AE is an open-set 3D object recognition method with a two-step structure, which is designed to identify objects of the out-of-the-distribution categories. We add an auto-encoder to generate the coarse embeddings after the first step.

- HGM$^2$R [7]: HGM$^2$R is a hypergraph-based 3DOR method specifically designed for the open-set environment, which assumes no distribution overlap between training and testing set. We construct an

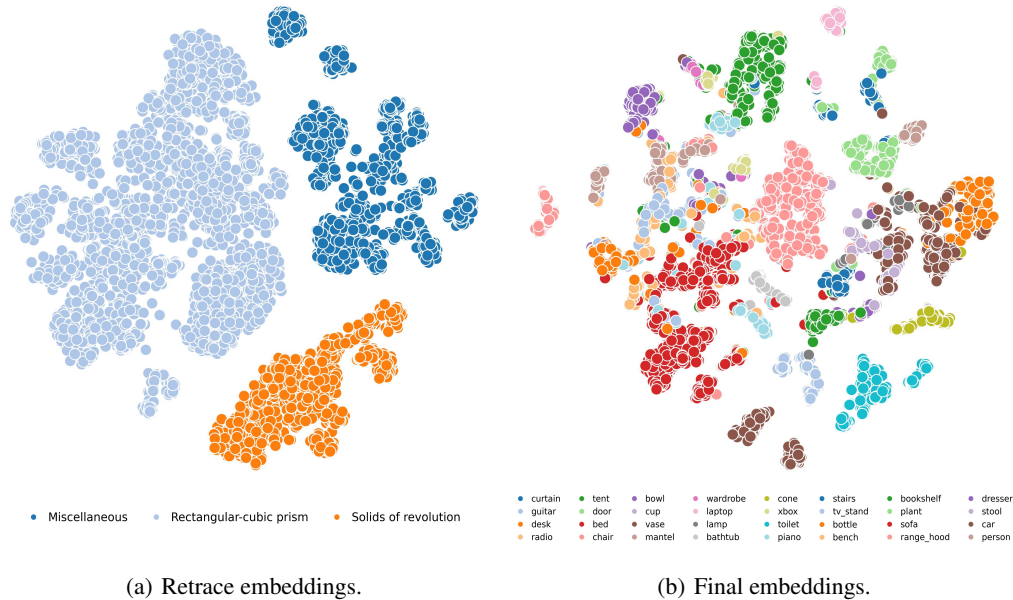

(a) Retrace embeddings.                    (b) Final embeddings.

Figure 6: The t-SNE visualization of the embeddings from unseen categories in the OS-MN40 dataset.

auto-encoder for coarse embedding after the MM3DOE module, and we construct only the knn-based hyperedges in the SAIKL module.

# E    Performance Result

The HRE module is designed to obtain multi-level embeddings of objects based on hierarchical categories. Therefore, we provide two visualized results of the retrace embeddings and final conditional embeddings of unseen categories in Figure 6(a) and Figure 6(b), respectively. These two t-SNE visualizations show that the proposed HRE and SET modules can fully leverage the hierarchical category information into multi-level retrace embedding and generalize them to unseen categories. This approach better captures the hierarchical semantic correlations in the wild and provides a practical framework for the representation learning of multi-label tasks towards a semi-open environment.
